# Object Recognition by Scene Alignment

**Bryan C. Russell    Antonio Torralba    Ce Liu    Rob Fergus    William T. Freeman**
Computer Science and Artificial Intelligence Laboratory
Massachusetts Institute of Technology
Cambridge, MA 02139 USA
{brussell,torralba,celiu,fergus,billf}@csail.mit.edu

## Abstract

Current object recognition systems can only recognize a limited number of object categories; scaling up to many categories is the next challenge. We seek to build a system to recognize and localize many different object categories in complex scenes. We achieve this through a simple approach: by matching the input image, in an appropriate representation, to images in a large training set of labeled images. Due to regularities in object identities across similar scenes, the retrieved matches provide hypotheses for object identities and locations. We build a probabilistic model to transfer the labels from the retrieval set to the input image. We demonstrate the effectiveness of this approach and study algorithm component contributions using held-out test sets from the LabelMe database.

## 1   Introduction

The recognition of objects in a scene often consists of matching representations of image regions to an object model while rejecting background regions. Recent examples of this approach include aligning pictorial cues [4], shape correspondence [1], and modeling the constellation of parts [5]. Other models, exploiting knowledge of the scene context in which the objects reside, have proven successful in boosting object recognition performance [18, 20, 15, 7, 13]. These methods model the relationship between scenes and objects and allow information transfer across the two.

Here, we exploit scene context using a different approach: we formulate the object detection problem as one of aligning elements of the entire scene to a large database of labeled images. The background, instead of being treated as a set of outliers, is used to guide the detection process. Our approach relies on the observation that when we have a large enough database of labeled images, we can find with high probability some images in the database that are very close to the query image in appearance, scene contents, and spatial arrangement [6, 19]. Since the images in the database are partially labeled, we can transfer the knowledge of the labeling to the query image. Figure 1 illustrates this idea. With these assumptions, the problem of object detection in scenes becomes a problem of aligning scenes. The main issues are: (1) Can we find a big enough dataset to span the required large number of scene configurations? (2) Given an input image, how do we find a set of images that aligns well with the query image? (3) How do we transfer the knowledge about objects contained in the labels?

The LabelMe dataset [14] is well-suited for this task, having a large number of images and labels spanning hundreds of object categories. Recent studies using non-parametric methods for computer vision and graphics [19, 6] show that when a large number of images are available, simple indexing techniques can be used to retrieve images with object arrangements similar to those of a query image.

The core part of our system is the transfer of labels from the images that best match the query image. We assume that there are commonalities amongst the labeled objects in the retrieved images and we cluster them to form candidate scenes. These scene clusters give hints as to what objects are depicted

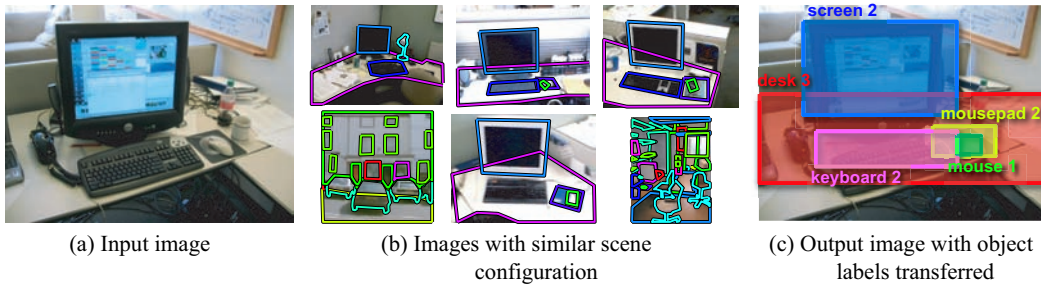

| (a) Input image | (b) Images with similar scene configuration | (c) Output image with object labels transferred |

Figure 1: Overview of our system. Given an input image, we search for images having a similar scene configuration in a large labeled database. The knowledge contained in the object labels for the best matching images is then transfered onto the input image to detect objects. Additional information, such as depth-ordering relationships between the objects, can also be transferred.

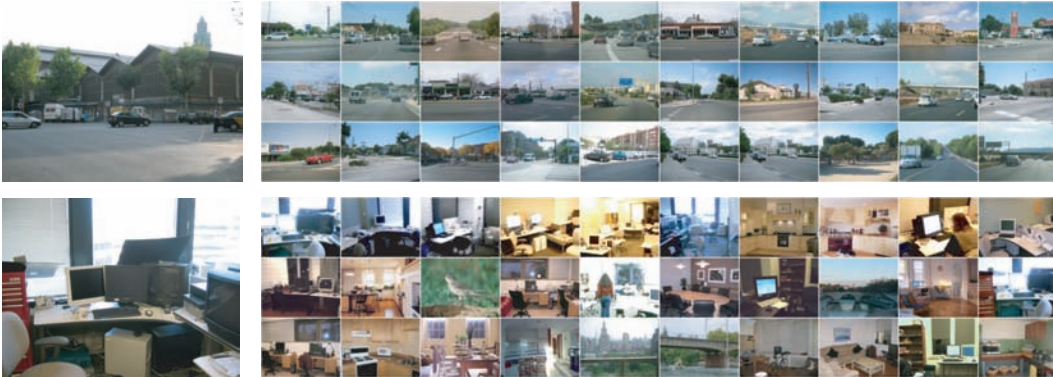

Figure 2: Retrieval set images. Each of the two rows depicts an input image (on the left) and 30 images from the LabelMe dataset [14] that best match the input image using the gist feature [12] and L1 distance (the images are sorted by their distances in raster order). Notice that the retrieved images generally belong to similar scene categories. Also the images contain mostly the same object categories, with the larger objects often matching in spatial location within the image. Many of the retrieved images share similar geometric perspective.

in the query image and their likely location. We describe a relatively simple generative model for determining which scene cluster best matches the query image and use this to detect objects.

The remaining sections are organized as follows: In Section 2, we describe our representation for scenes and objects. We formulate a model that integrates the information in the object labels with object detectors in Section 3. In Section 4, we extend this model to allow clustering of the retrieved images based on the object labels. We show experimental results of our system output in Section 5, and conclude in Section 6.

## 2 Matching Scenes and Objects with the Gist Feature

We describe the gist feature [12], which is a low dimensional representation of an image region and has been shown to achieve good performance for the scene recognition task when applied to an entire image. To construct the gist feature, an image region is passed through a Gabor filter bank comprising 4 scales and 8 orientations. The image region is divided into a 4x4 non-overlapping grid and the output energy of each filter is averaged within each grid cell. The resulting representation is a $4 \times 8 \times 16 = 512$ dimensional vector. Note that the gist feature preserves spatial structure information and is similar to applying the SIFT descriptor [9] to the image region.

We consider the task of retrieving a set of images (which we refer to as the *retrieval set*) that closely matches the scene contents and geometrical layout of an input image. Figure 2 shows retrieval sets for two typical input images using the gist feature. We show the top 30 closest matching images from the LabelMe database based on the L1-norm distance, which is robust to outliers. Notice that the gist feature retrieves images that match the scene type of the input image. Furthermore, many of the objects depicted in the input image appear in the retrieval set, with the larger objects residing in approximately the same spatial location relative to the image. Also, the retrieval set has many

images that share a similar geometric perspective. Of course, not every retrieved image matches well and we account for outliers in Section 4.

We evaluate the ability of the retrieval set to predict the presence of objects in the input image. For this, we found a retrieval set of 200 images and formed a normalized histogram (the histogram entries sum to one) of the object categories that were labeled. We compute performance for object categories with at least 200 training examples and that appear in at least 15 test images. We compute the area under the ROC curve for each object category. As a comparison, we evaluate the performance of an SVM applied to gist features by using the maximal score over a set of bounding boxes extracted from the image. The area under ROC performance of the retrieval set versus the SVM is shown in Figure 3 as a scatter plot, with each point corresponding to a tested object category. As a guide, a diagonal line is displayed; those points that reside above the diagonal indicate better SVM performance (and vice versa). Notice that the retrieval set predicts well the objects present in the input image and outperforms the detectors based on local appearance information (the SVM) for most object classes.

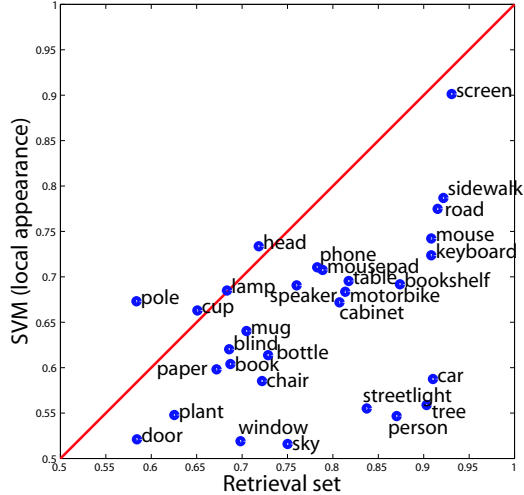

Figure 3: Evaluation of the goodness of the retrieval set by how well it predicts which objects are present in the input image. We build a simple classifier based on object counts in the retrieval set as provided by their associated LabelMe object labels. We compare this to detection based on local appearance alone using an SVM applied to bounding boxes in the input image (the maximal score is used). The area under the ROC curve is computed for many object categories for the two classifiers. Performance is shown as a scatter plot where each point represents an object category. Notice that the retrieval set predicts well object presence and in a majority cases outperforms the SVM output, which is based only on local appearance.

# 3  Utilizing Retrieval Set Images for Object Detection

In Section 2, we observed that the set of labels corresponding to images that best match an input image predict well the contents of the input image. In this section, we will describe a model that integrates local appearance with object presence and spatial likelihood information given by the object labels belonging to the retrieval set.

We wish to model the relationship between object categories $o$, their spatial location $x$ within an image, and their appearance $g$. For a set of $N$ images, each having $M_i$ object proposals over $L$ object categories, we assume a joint model that factorizes as follows:

$$p(o,x,g|\theta,\phi,\eta) = \prod_{i=1}^{N}\prod_{j=1}^{M_i}\sum_{h_{i,j}=0}^{1} p(o_{i,j}|h_{i,j},\theta)\, p(x_{i,j}|o_{i,j},h_{i,j},\phi)\, p(g_{i,j}|o_{i,j},h_{i,j},\eta) \qquad (1)$$

We assume that the joint model factorizes as a product of three terms: (i) $p(o_{i,j}|h_{i,j}=m,\theta_m)$, the likelihood of which object categories will appear in the image, (ii) $p(x_{i,j}|o_{i,j}=l,h_{i,j}=m,\phi_{m,l})$, the likely spatial locations of observing object category $l$ in the image, and (iii) $p(g_{i,j}|o_{i,j}=l,h_{i,j}=m,\eta_{m,l})$, the appearance likelihood of object category $l$. We let $h_{i,j}=1$ indicate whether object category $o_{i,j}$ is actually present in location $x_{i,j}$ ($h_{i,j}=0$ indicates absence). Figure 4 depicts the above as a graphical model. We use plate notation, where the variable nodes inside a plate are duplicated based on the counts depicted in the top-left corner of the plate.

We instantiate the model as follows. The spatial location of objects are parameterized as bounding boxes $x_{i,j}=(c_{i,j}^x, c_{i,j}^y, c_{i,j}^w, c_{i,j}^h)$ where $(c_{i,j}^x, c_{i,j}^y)$ is the centroid and $(c_{i,j}^w, c_{i,j}^w)$ is the width and

height (bounding boxes are extracted from object labels by tightly cropping the polygonal annotation). Each component of $x_{i,j}$ is normalized with respect to the image to lie in $[0, 1]$. We assume $\theta_m$ are multinomial parameters and $\phi_{m,l} = (\mu_{m,l}, \Lambda_{m,l})$ are Gaussian means and covariances over the bounding box parameters. Finally, we assume $g_{i,j}$ is the output of a trained SVM applied to a gist feature $\tilde{g}_{i,j}$. We let $\eta_{m,l}$ parameterize the logistic function $(1 + \exp(-\eta_{m,l} [1\ g_{i,j}]^T))^{-1}$.

The parameters $\eta_{m,l}$ are learned offline by first training SVMs for each object class on the set of all labeled examples of object class $l$ and a set of distractors. We then fit logistic functions to the positive and negative examples of each class. We learn the parameters $\theta_m$ and $\phi_{m,l}$ online using the object labels corresponding to the retrieval set. These are learned by simply counting the object class occurrences and fitting Gaussians to the bounding boxes corresponding to the object labels.

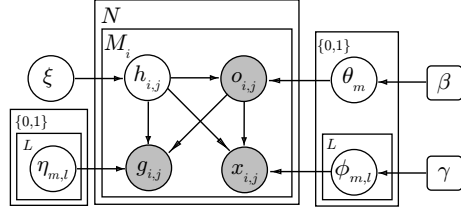

Figure 4: Graphical model that integrates information about which objects are likely to be present in the image $o$, their appearance $g$, and their likely spatial location $x$. The parameters for object appearance $\eta$ are learned offline using positive and negative examples for each object class. The parameters for object presence likelihood $\theta$ and spatial location $\phi$ are learned online from the retrieval set. For all possible bounding boxes in the input image, we wish to infer $h$, which indicates whether an object is present or absent.

For the input image, we wish to infer the latent variables $h_{i,j}$ corresponding to a dense sampling of all possible bounding box locations $x_{i,j}$ and object classes $o_{i,j}$ using the learned parameters $\theta_m$, $\phi_{m,l}$, and $\eta_{m,l}$. For this, we compute the postierior distribution $p(h_{i,j} = m | o_{i,j} = l, x_{i,j}, g_{i,j}, \theta_m, \phi_{m,l}, \eta_{m,l})$, which is proportional to the product of the three learned distributions, for $m = \{0, 1\}$.

The procedure outlined here allows for significant computational savings over naive application of an object detector. Without finding similar images that match the input scene configuration, we would need to apply an object detector densely across the entire image for all object categories. In contrast, our model can constrain which object categories to look for and where. More precisely, we only need to consider object categories with relatively high probability in the scene model and bounding boxes within the range of the likely search locations. These can be decided based on thresholds. Also note that the conditional independences implied by the graphical model allows us to fit the parameters from the retrieval set and train the object detectors separately.

Note that for tractability, we assume Dirichlet and Normal-Inverse-Wishart conjugate prior distributions over $\theta_m$ and $\phi_{m,l}$ with hyperparemters $\beta$ and $\gamma = (\kappa, \vartheta, \nu, \Delta)$ (expected mean $\vartheta$, $\kappa$ pseudo-counts on the scale of the spatial observations, $\nu$ degrees of freedom, and sample covariance $\Delta$). Furthermore, we assume a Bernoulli prior distribution over $h_{i,j}$ parameterized by $\xi = 0.5$. We hand-tuned the remaining parameters in the model. For $h_{i,j} = 0$, we assume the noninformative distributions $o_{i,j} \sim Uniform(1/L)$ and each component of $x_{i,j} \sim Uniform(1)$.

# 4 Clustering Retrieval Set Images for Robustness to Mismatches

While many images in the retrieval set match the input image scene configuration and contents, there are also outliers. Typically, most of the labeled objects in the outlier images are not present in the input image or in the set of correctly matched retrieval images. In this section, we describe a process to organize the retrieval set images into consistent clusters based on the co-occurrence of the object labels within the images. The clusters will typically correspond to different scene types and/or viewpoints. The task is to then automatically choose the cluster of retrieval set images that will best assist us in detecting objects in the input image.

We augment the model of Section 3 by assigning each image to a latent cluster $s_i$. The cluster assignments are distributed according to the mixing weights $\pi$. We depict the model in Figure 5(a). Intuitively, the model finds clusters using the object labels $o_{i,j}$ and their spatial location $x_{i,j}$ within the retrieved set of images. To automatically infer the number of clusters, we use a Dirichlet Process prior on the mixing weights $\pi \sim Stick(\alpha)$, where $Stick(\alpha)$ is the stick-breaking process of Grif-

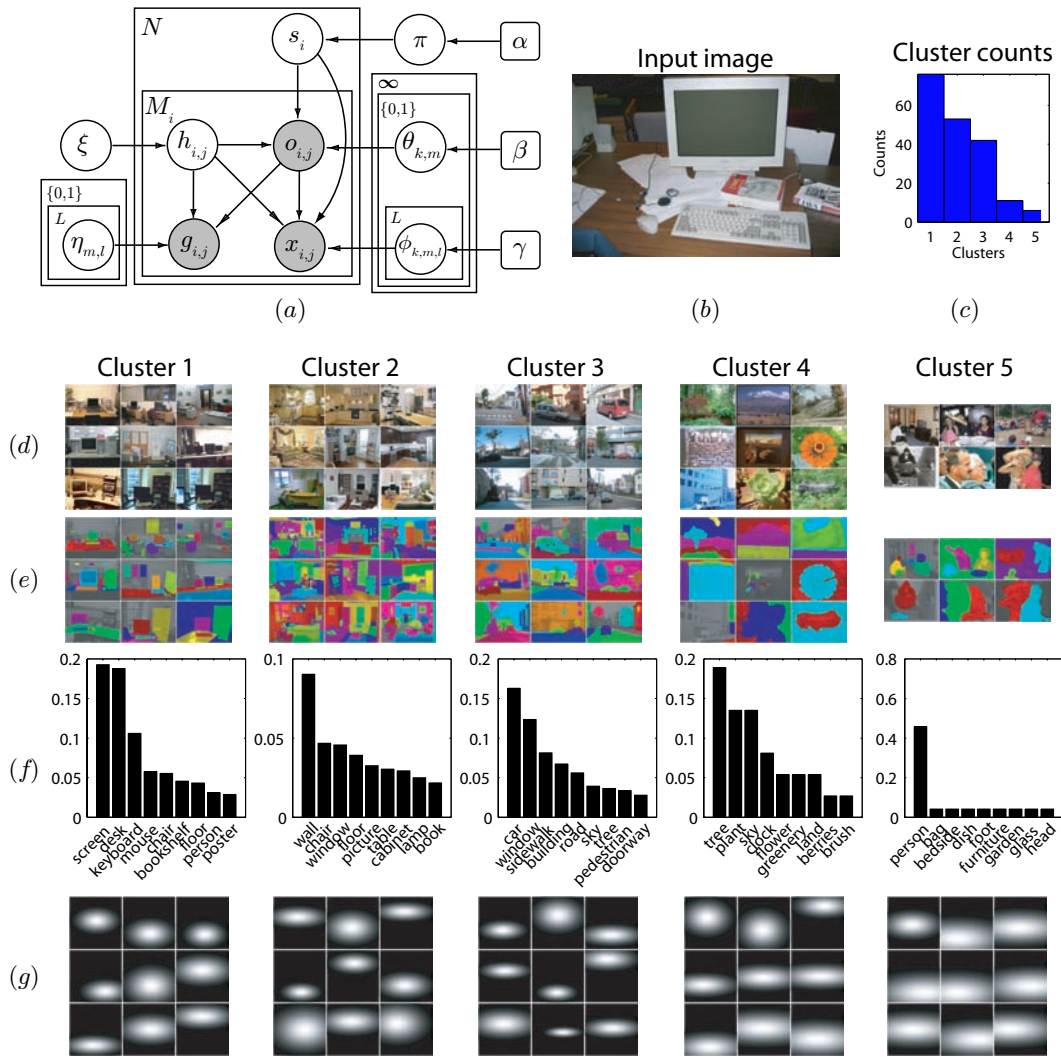

Figure 5: (a) Graphical model for clustering retrieval set images using their object labels. We extend the model of Figure 4 to allow each image to be assigned to a latent cluster $s_i$, which is drawn from mixing weights $\pi$. We use a Dirichlet process prior to automatically infer the number of clusters. We illustrate the clustering process for the retrieval set corresponding to the input image in (b). (c) Histogram of the number of images assigned to the five clusters with highest likelihood. (d) Montages of retrieval set images assigned to each cluster, along with their object labels (colors show spatial extent), shown in (e). (f) The likelihood of an object category being present in a given cluster (the top nine most likely objects are listed). (g) Spatial likelihoods for the objects listed in (f). Note that the montage cells are sorted in raster order.

fiths, Engen, and McCloskey [8, 11, 16] with concentration parameter $\alpha$. In the Chinese restaurant analogy, the different clusters correspond to tables and the parameters for object presence $\theta_k$ and spatial location $\phi_k$ are the dishes served at a given table. An image (along with its object labels) corresponds to a single customer that is seated at a table.

We illustrate the clustering process for a retrieval set belonging to the input image in Figure 5(b). The five clusters with highest likelihood are visualized in the columns of Figure 5(d)-(g). Figure 5(d) shows montages of retrieval images with highest likelihood that were assigned to each cluster. The total number of retrieval images that were assigned to each cluster are shown as a histogram in Figure 5(c). The number of images assigned to each cluster is proportional to the cluster mixing weights, $\pi$. Figure 5(e) depicts the object labels that were provided for the images in Figure 5(d), with the colors showing the spatial extent of the object labels. Notice that the images and labels belonging to each cluster share approximately the same object categories and geometrical configuration. Also, the cluster that best matches the input image tends to have the highest number of retrieval images assigned to it. Figure 5(f) shows the likelihood of objects that appear in the cluster

(the nine objects with highest likelihood are shown). This corresponds to $\theta$ in the model. Figure 5(g) depicts the spatial distribution of the object centroid within the cluster. The montage of nine cells correspond to the nine objects listed in Figure 5(f), sorted in raster order. The spatial distributions illustrate $\phi$. Notice that typically at least one cluster predicts well the objects contained in the input image, in addition to their location, via the object likelihoods and spatial distributions.

To learn $\theta_k$ and $\phi_k$, we use a Rao-Blackwellized Gibbs sampler to draw samples from the posterior distribution over $s_i$ given the object labels belonging to the set of retrieved images. We ran the Gibbs sampler for 100 iterations. Empirically, we observed relatively fast convergence to a stable solution. Note that improved performance may be achieved with variational inference for Dirichlet Processes [10, 17]. We manually tuned all hyperparameters using a validation set of images, with concentration parameter $\alpha = 100$ and spatial location parameters $\kappa = 0.1$, $\vartheta = 0.5$, $\nu = 3$, and $\Delta = 0.01$ across all bounding box parameters (with the exception of $\Delta = 0.1$ for the horizontal centroid location, which reflects less certainty a priori about the horizontal location of objects). We used a symmetric Dirichlet hyperparameter with $\beta_l = 0.1$ across all object categories $l$.

For final object detection, we use the learned parameters $\pi$, $\theta$, and $\phi$ to infer $h_{i,j}$. Since $s_i$ and $h_{i,j}$ are latent random variables for the input image, we perform hard EM by marginalizing over $h_{i,j}$ to infer the best cluster $s_i^*$. We then in turn fix $s_i^*$ and infer $h_{i,j}$, as outlined in Section 3.

# 5    Experimental Results

In this section we show qualitative and quantitative results for our model. We use a subset of the LabelMe dataset for our experiments, discarding spurious and nonlabeled images. The dataset is split into training and test sets. The training set has 15691 images and 105034 annotations. The test set has 560 images and 3571 annotations. The test set comprises images of street scenes and indoor office scenes. To avoid overfitting, we used street scene images that were photographed in a different city from the images in the training set. To overcome the diverse object labels provided by users of LabelMe, we used WordNet [3] to resolve synonyms. For object detection, we extracted 3809 bounding boxes per image. For the final detection results, we used non-maximal suppression.

Example object detections from our system are shown in Figure 6(b),(d),(e). Notice that our system can find many different objects embedded in different scene type configurations. When mistakes are made, the proposed object location typically makes sense within the scene. In Figure 6(c), we compare against a baseline object detector using only appearance information and trained with a linear kernel SVM. Thresholds for both detectors were set to yield a 0.5 false positive rate per image for each object category ($\sim$1.3e-4 false positives per window). Notice that our system produces more detections and rejects objects that do not belong to the scene. In Figure 6(e), we show typical failures of the system, which usually occurs when the retrieval set is not correct or an input image is outside of the training set.

In Figure 7, we show quantitative results for object detection for a number of object categories. We show ROC curves (plotted on log-log axes) for the local appearance detector, the detector from Section 3 (without clustering), and the full system with clustering. We scored detections using the PASCAL VOC 2006 criteria [2], where the outputs are sorted from most confident to least and the ratio of intersection area to union area is computed between an output bounding box and ground-truth bounding box. If the ratio exceeds 0.5, then the output is deemed correct and the ground-truth label is removed. While this scoring criteria is good for some objects, other objects are not well represented by bounding boxes (e.g. buildings and sky).

Notice that the detectors that take into account context typically outperforms the detector using local appearance only. Also, clustering does as well and in some cases outperforms no clustering. Finally, the overall system sometimes performs worse for indoor scenes. This is due to poor retrieval set matching, which causes a poor context model to be learned.

# 6    Conclusion

We presented a framework for object detection in scenes based on transferring knowledge about objects from a large labeled image database. We have shown that a relatively simple parametric

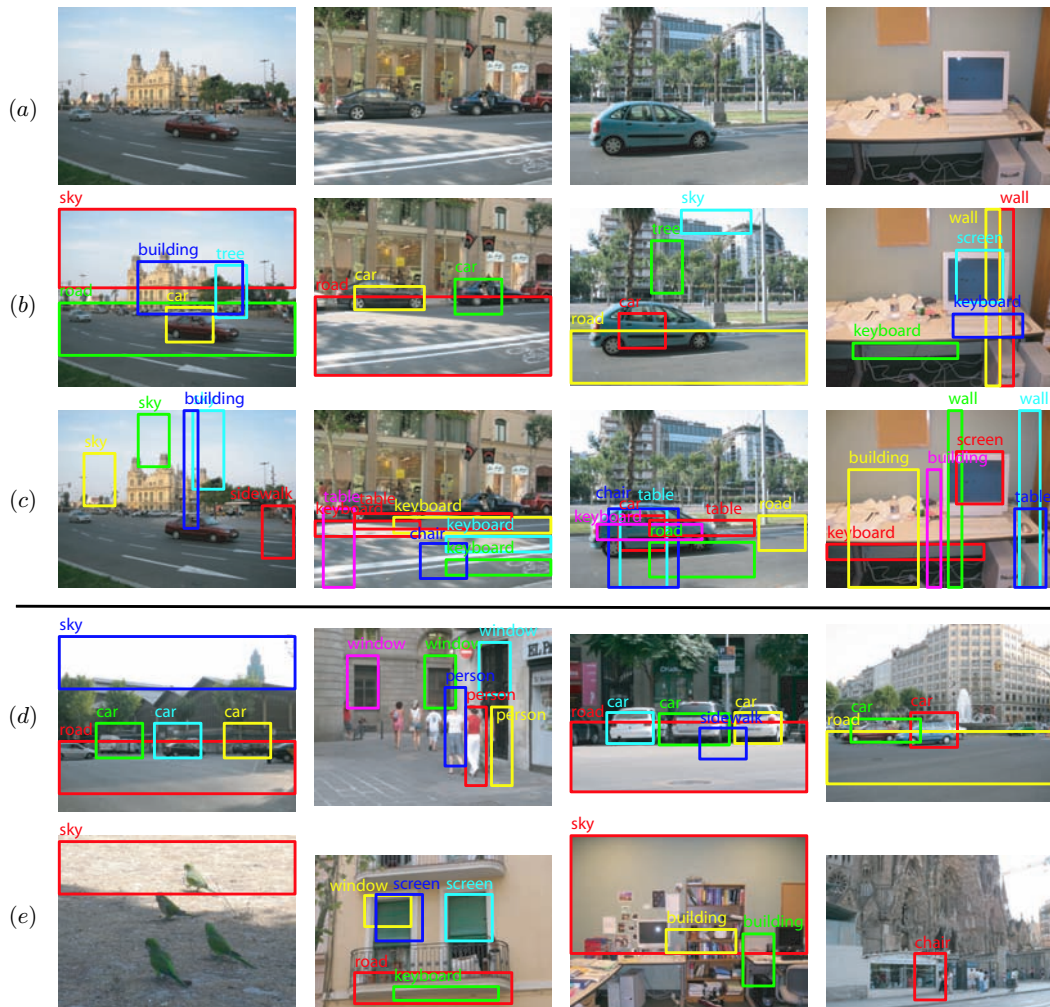

Figure 6: (a) Input images. (b) Object detections from our system combining scene alignment with local detection. (c) Object detections using appearance information only with an SVM. Notice that our system detects more objects and rejects out-of-context objects. (d) More outputs from our system. Notice that many different object categories are detected across different scenes. (e) Failure cases for our system. These often occur when the retrieval set is incorrect.

model, trained on images loosely matching the spatial configuration of the input image, is capable of accurately inferring which objects are depicted in the input image along with their location. We showed that we can successfully detect a wide range of objects depicted in a variety of scene types.

# 7  Acknowledgments

This work was supported by the National Science Foundation Grant No. 0413232, the National Geospatial-Intelligence Agency NEGI-1582-04-0004, and the Office of Naval Research MURI Grant N00014-06-1-0734.

# References

[1] A. Berg, T. Berg, and J. Malik. Shape matching and object recognition using low distortion correspondence. In *CVPR*, volume 1, pages 26–33, June 2005.

[2] M. Everingham, A. Zisserman, C.K.I. Williams, and L. Van Gool. The pascal visual object classes challenge 2006 (voc 2006) results. Technical report, September 2006. The PASCAL2006 dataset can be downloaded at http : //www.pascal − network.org/challenges/VOC/voc2006/.

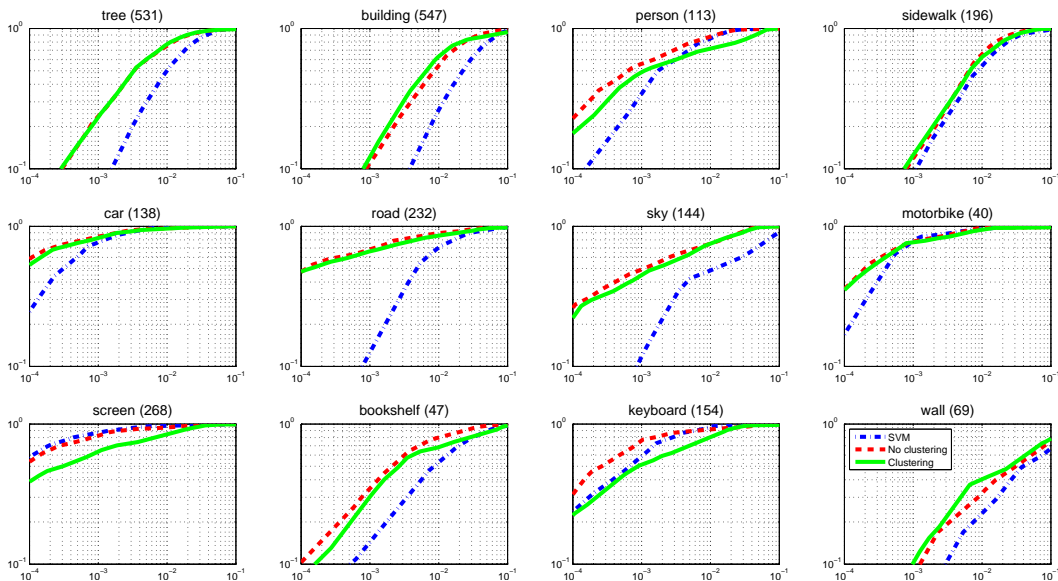

Figure 7: Comparison of full system against local appearance only detector (SVM). Detection rate for a number of object categories tested at a fixed false positive per window rate of 2e-04 (0.8 false positives per image per object class). The number of test examples appear in parenthesis next to the category name. We plot performance for a number of classes for the baseline SVM object detector (blue), the detector of Section 3 using no clustering (red), and the full system (green). Notice that detectors taking into account context performs better in most cases than using local appearance alone. Also, clustering does as well, and sometimes exceeds no clustering. Notable exceptions are for some indoor object categories. This is due to poor retrieval set matching, which causes a poor context model to be learned.

[3] C. Fellbaum. *Wordnet: An Electronic Lexical Database*. Bradford Books, 1998.

[4] P. Felzenszwalb and D. Huttenlocher. Pictorial structures for object recognition. *Intl. J. Computer Vision*, 61(1), 2005.

[5] R. Fergus, P. Perona, and A. Zisserman. Object class recognition by unsupervised scale-invariant learning. In *CVPR*, 2003.

[6] James Hays and Alexei Efros. Scene completion using millions of photographs. In *"SIGGRAPH"*, 2007.

[7] D. Hoiem, A. Efros, and M. Hebert. Putting objects in perspective. In *CVPR*, 2006.

[8] H. Ishwaran and M. Zarepour. Exact and approximate sum-representations for the dirichlet process. *Canadian Journal of Statistics*, 30:269–283, 2002.

[9] David G. Lowe. Distinctive image features from scale-invariant keypoints. *Intl. J. Computer Vision*, 60(2):91–110, 2004.

[10] J. McAuliffe, D. Blei, and M. Jordan. Nonparametric empirical bayes for the Dirichlet process mixture model. *Statistics and Computing*, 16:5–14, 2006.

[11] R. M. Neal. Density modeling and clustering using Dirichlet diffusion trees. *In Bayesian Statistics*, 7:619–629, 2003.

[12] A. Oliva and A. Torralba. Modeling the shape of the scene: a holistic representation of the spatial envelope. *Intl. J. Computer Vision*, 42(3):145–175, 2001.

[13] A. Rabinovich, A. Vedaldi, C. Galleguillos, E. Wiewiora, and S. Belongie. Objects in context. In *IEEE Intl. Conf. on Computer Vision*, 2007.

[14] B. C. Russell, A. Torralba, K. P. Murphy, and W. T. Freeman. Labelme: a database and web-based tool for image annotation. Technical Report AIM-2005-025, MIT AI Lab Memo, September, 2005.

[15] E. Sudderth, A. Torralba, W. T. Freeman, and W. Willsky. Learning hierarchical models of scenes, objects, and parts. In *IEEE Intl. Conf. on Computer Vision*, 2005.

[16] Y. W. Teh, M. I. Jordan, M. J. Beal, and D. M. Blei. Hierarchical Dirichlet processes. *Journal of the American Statistical Association*, 2006.

[17] Y. W. Teh, D. Newman, and Welling M. A collapsed variational bayesian inference algorithm for latent dirichlet allocation. In *Advances in Neural Info. Proc. Systems*, 2006.

[18] A. Torralba. Contextual priming for object detection. *Intl. J. Computer Vision*, 53(2):153–167, 2003.

[19] A. Torralba, R. Fergus, and W.T. Freeman. Tiny images. Technical Report AIM-2005-025, MIT AI Lab Memo, September, 2005.

[20] A. Torralba, K. Murphy, W. Freeman, and M. Rubin. Context-based vision system for place and object recognition. In *Intl. Conf. Computer Vision*, 2003.

